# Sufficient Conditions for Generating Group Level Sparsity in a Robust Minimax Framework

**Hongbo Zhou and Qiang Cheng**
Computer Science department,
Southern Illinois University Carbondale, IL, 62901
hongboz@siu.edu, qcheng@cs.siu.edu

## Abstract

Regularization technique has become a principled tool for statistics and machine learning research and practice. However, in most situations, these regularization terms are not well interpreted, especially on how they are related to the loss function and data. In this paper, we propose a robust minimax framework to interpret the relationship between data and regularization terms for a large class of loss functions. We show that various regularization terms are essentially corresponding to different distortions to the original data matrix. This minimax framework includes ridge regression, lasso, elastic net, fused lasso, group lasso, local coordinate coding, multiple kernel learning, etc., as special cases. Within this minimax framework, we further give mathematically exact definition for a novel representation called sparse grouping representation (SGR), and prove a set of sufficient conditions for generating such group level sparsity. Under these sufficient conditions, a large set of consistent regularization terms can be designed. This SGR is essentially different from group lasso in the way of using class or group information, and it outperforms group lasso when there appears group label noise. We also provide some generalization bounds in a classification setting.

## 1 Introduction

A general form of estimating a quantity $w \in \mathcal{R}^n$ from an empirical measurement set $\mathcal{X}$ by minimizing a regularized or penalized functional is

$$\hat{w} = \underset{w}{\operatorname{argmin}}\{\mathcal{L}(I_w(\mathcal{X})) + \lambda \mathcal{J}(w)\}, \tag{1}$$

where $I_w(\mathcal{X}) \in \mathcal{R}^m$ expresses the relationship between $w$ and data $\mathcal{X}$, $\mathcal{L}(.) := \mathcal{R}^m \to \mathcal{R}^+$ is a loss function, $\mathcal{J}(.) := \mathcal{R}^n \to \mathcal{R}^+$ is a regularization term and $\lambda \in \mathcal{R}$ is a weight. Positive integers $n, m$ represent the dimensions of the associated Euclidean spaces. Varying in specific applications, the loss function $\mathcal{L}$ has lots of forms, and the most often used are these induced (A is induced by B, means B is the core part of A) by squared Euclidean norm or squared Hilbertian norms. Empirically, the functional $\mathcal{J}$ is often interpreted as smoothing function, model bias or uncertainty. Although Equation (1) has been widely used, it is difficult to establish a general mathematically exact relationship between $\mathcal{L}$ and $\mathcal{J}$. This directly encumbers the interpretability of parameters in the model selection. It would be desirable if we can represent Equation (1) by a simpler form

$$\hat{w} = \underset{w}{\operatorname{argmin}} \mathcal{L}'(I_w'(\mathcal{X})). \tag{2}$$

Obviously, Equation (2) provides a better interpretability for the regularization term in Equation (1) by explicitly expressing the model bias or uncertainty as a variable of the relationship functional. In this paper, we introduce a minimax framework and show that for a large family of Euclidean norm induced loss functions, an equivalence relationship between Equation (1) and Equation (2) can be

established. Moreover, the model bias or uncertainty will be expressed as distortions associated with certain functional spaces. We will give a series of corollaries to show that well-studied lasso, group lasso, local coordinate coding, multiple kernel learning, etc., are all special cases of this novel framework. As a result, we shall see that various regularization terms associated with lasso, group lasso, etc., can be interpreted as distortions that belong to different distortion sets.

Within this framework, we further investigate a large family of distortion sets which can generate a special type of group level sparsity which we call sparse grouping representation (SGR). Instead of merely designing one specific regularization term, we give sufficient conditions for the distortion sets to generate the SGR. Under these sufficient conditions, a large set of consistent regularization terms can be designed. Compared with the well-known group lasso which uses group distribution information in a supervised learning setting, the SGR is an unsupervised one and thus essentially different from the group lasso. In a novel fault-tolerance classification application, where there appears class or group label noise, we show that the SGR outperforms the group lasso. This is not surprising because the class or group label information is used as a core part of the group lasso while the group sparsity produced by the SGR is intrinsic, in that the SGR does not need the class label information as priors. Finally, we also note that the group level sparsity is of great interests due to its wide applications in various supervised learning settings.

In this paper, we will state our results in a classification setting. In Section 2 we will review some closely related work, and we will introduce the robust minimax framework in Section 3. In Section 4, we will define the sparse grouping representation and prove a set of sufficient conditions for generating group level sparsity. An experimental verification on a low resolution face recognition task will be reported in Section 5.

## 2   Related Work

In this paper, we will mainly work with the penalized linear regression problem and we shall review some closely related work here. For penalized linear regression, several well-studied regularization procedures are ridge regression or Tikhonov regularization [15], bridge regression [10], lasso [19] and subset selection [5], fused lasso [20], elastic net [27], group lasso [25], multiple kernel learning [3, 2], local coordinate coding [24], etc. The lasso has at least three prominent features to make itself a principled tool among all of these procedures: continuous shrinkage and automatic variable selection at the same time, computational tractability (can be solved by linear programming methods) as well as inducing sparsity. Recent results show that lasso can recover the solution of $l_0$ regularization under certain regularity conditions [8, 6, 7]. Recent advances such as fused lasso [20], elastic net [27], group lasso [25] and local coordinate coding [24] are motivated by lasso [19].

Two concepts closely related to our work are the elastic net or grouping effect observed by [27] and the group lasso [25]. The elastic net model hybridizes lasso and ridge regression to preserve some redundancy for the variable selection, and it can be viewed as a stabilized version of lasso [27] and hence it is still biased. The group lasso can produce group level sparsity [25, 2] but it requires the group label information as prior. We shall see that in a novel classification application when there appears class label noise [22, 18, 17, 26], the group lasso fails. We will discuss the differences of various regularization procedures in a classification setting. We will use the basic schema for the sparse representation classification (SRC) algorithm proposed in [21], and different regularization procedures will be used to replace the lasso in the SRC.

The proposed framework reveals a fundamental connection between robust linear regression and various regularized techniques using regularization terms of $l_0$, $l_1$, $l_2$, etc. Although [11] first introduced a robust model for least square problem with uncertain data and [23] discussed a robust model for lasso, our results allow for using any positive regularization functions and a large family of loss functions.

## 3   Minimax Framework for Robust Linear Regression

In this section, we will start with taking the loss function $\mathcal{L}$ as squared Euclidean norm, and we will generalize the results to other loss functions in section 3.4.

## 3.1 Notations and Problem Statement

In a general $M$ $(M > 1)$-classes classification setting, we are given a training dataset $\mathbf{T} = \{(x_i, g_i)\}_{i=1}^n$, where $x_i \in \mathcal{R}^p$ is the feature vector and $g_i \in \{1, \cdots, M\}$ is the class label for the $i$th observation. A data (observation) matrix is formed as $\mathbf{A} = [x_1, \cdots, x_n]$ of size $p \times n$. Given a test example $y$, the goal is to determine its class label.

## 3.2 Distortion Models

Assume that the $j$th class $C_j$ has $n_j$ observations $x_1^{(j)}, \cdots, x_{n_j}^{(j)}$. If $x$ belongs to the $j$th class, then $x \in \text{span}\{x_1^{(j)}, \cdots, x_{n_j}^{(j)}\}$. We approximate $y$ by a linear combination of the training examples:

$$y = Aw + \eta, \tag{3}$$

where $w = [w_1, w_2, \cdots, w_n]^T$ is a vector of combining coefficients; and $\eta \in \mathcal{R}^p$ represents a vector of additive zero-mean noise. We assume a Gaussian model $v \sim N(0, \sigma^2 I)$ for this additive noise, so a least squares estimator can be used to compute the combining coefficients.

The observed training dataset $\mathbf{T}$ may have undergone various noise or distortions. We define the following two classes of distortion models.

**Definition 1:** *A random matrix $\Delta A$ is called bounded example-wise (or attribute) distortion $(BED)$ with a bound $\lambda$, denoted as $BED(\lambda)$, if $\Delta A := [d_1, \cdots, d_n]$, $d_k \in \mathcal{R}^p$, $||d_k||_2 \leq \lambda, k = 1, \cdots, n$. where $\lambda$ is a positive parameter.*

This distortion model assumes that each observation (signal) is distorted independently from the other observations, and the distortion has a uniformly upper bounded energy ("uniformity" refers to the fact that all the examples have the same bound). $BED$ includes attribute noise defined in [22, 26], and some examples of $BED$ include Gaussian noise and sampling noise in face recognition.

**Definition 2:** *A random matrix $\Delta A$ is called bounded coefficient distortion $(BCD)$ with bound $f$, denoted as $BCD(f)$, if $||\Delta Aw||_2 \leq f(w), \forall w \in \mathcal{R}^p$, where $f(w) \in R^+$ .*

The above definition allows for any distortion with or without inter-observation dependency. For example, we can take $f(w) = \lambda ||w||_2$, and Definition 2 with this $f(w)$ means that the maximum eigenvalue of $\Delta A$ is upper limited by $\lambda$. This can be easily seen as follows. Denote the maximum eigenvalue of $\Delta A$ by $\sigma_{max}(\Delta A)$. Then we have

$$\sigma_{max}(\Delta A) = \sup_{u,v \neq 0} \frac{u^T \Delta Av}{||u||_2 ||v||_2} = \sup_{u \neq 0} \frac{||\Delta Au||_2}{||u||_2},$$

which is a standard result from the singular value decomposition (SVD) [12]. That is, the condition of $||\Delta Aw||_2 \leq \lambda ||w||_2$ is equivalent to the condition that the maximum eigenvalue of $\Delta A$ is upper bounded by $\lambda$. In fact, $BED$ is a subset of $BCD$ by using triangular inequality and taking special forms of $f(w)$. We will use $\mathcal{D} := BCD$ to represent the distortion model.

Besides the additive residue $\eta$ generated from fitting models, to account for the above distortion models, we shall consider multiplicative noise by extending Equation (3) as follows:

$$y = (A + \Delta A)w + \eta, \tag{4}$$

where $\Delta A \in \mathcal{D}$ represents a possible distortion imposed to the observations.

## 3.3 Fundamental Theorem of Distortion

Now with the above refined linear model that incorporates a distortion model, we estimate the model parameters $w$ by minimizing the variance of Gaussian residues for the worst distortions within a permissible distortion set $\mathcal{D}$. Thus our robust model is

$$\min_{w \in \mathcal{R}^p} \max_{\Delta A \in \mathcal{D}} ||y - (A + \Delta A)w||_2. \tag{5}$$

The above minimax estimation will be used in our robust framework.

An advantage of this model is that it considers additive noise as well as multiplicative one within a class of allowable noise models. As the optimal estimation of the model parameter in Equation

(5), $w^*$, is derived for the worst distortion in $\mathcal{D}$, $w^*$ will be insensitive to any deviation from the underlying (unknown) noise-free examples, provided the deviation is limited to the tolerance level given by $\mathcal{D}$. The estimate $w^*$ thus is applicable to any $A + \Delta A$ with $\Delta A \in \mathcal{D}$. In brief, the robustness of our framework is offered by modeling possible multiplicative noise as well as the consequent insensitivity of the estimated parameter to any deviations (within $\mathcal{D}$) from the noise-free underlying (unknown) data. Moreover, this model can seamlessly incorporate either example-wise noise or class noise, or both.

Equation (5) provides a clear interpretation of the robust model. In the following, we will give a theorem to show an equivalence relationship between the robust minimax model of Equation (5) and a general form of regularized linear regression procedure.

**Theorem 1.** *Equation (5) with distortion set $\mathcal{D}(f)$ is equivalent to the following generalized regularized minimization problem:*

$$\min_{w \in \mathcal{R}^p} ||y - Aw||_2 + f(w). \tag{6}$$

Sketch of the proof: Fix $w = w^*$ and establish equality between upper bound and lower bound.

$$||y - (A + \Delta A)w^*||_2 \leq ||y - Aw^*||_2 + ||\Delta A w^*||_2$$
$$\leq ||y - Aw^*||_2 + f(w^*).$$

In the above we have used the triangle inequality of norms. If $y - Aw^* \neq 0$, we define $u = (y - Aw^*)/||y - Aw^*||_2$. Since $\max_{\Delta A \in \mathcal{D}} f(\Delta A) \geq f(\Delta A^*)$, by taking $\Delta A^* = -u f(w^*) t(w^*)^T / k$, where $t(w_i^*) = 1/w_i^*$ for $w_i^* \neq 0$, $t(w_i^*) = 0$ for $w_i^* = 0$ and $k$ is the number of non-zero $w_i^*$ (note that $w^*$ is fixed so we can define $t(w^*)$), we can actually attain the upper bound. It is easily verified that the expression is also valid if $y - Aw^* = 0$.

Theorem 1 gives an equivalence relationship between general regularized least squares problems and the robust regression under certain distortions. It should be noted that Equation (6) involves $\min ||.||_2$, and the standard form for least squares problem uses $\min ||.||_2^2$ as a loss function. It is known that these two coincide up to a change of the regularization coefficient so the following conclusions are valid for both of them. Several corollaries related to $l_0, l_1, l_2$, elastic net, group lasso, local coordinate coding, etc., can be derived based on Theorem 1.

**Corollary 1:** *$l_0$ regularized regression is equivalent to taking a distortion set $\mathcal{D}(f^{l_0})$ where $f^{l_0}(w) = t(w)w^T$, $t(w_i) = 1/w_i$ for $w_i \neq 0$, $t(w_i) = 0$ for $w_i = 0$.*

**Corollary 2:** *$l_1$ regularized regression (lasso) is equivalent to taking a distortion set $\mathcal{D}(f^{l_1})$ where $f^{l_1}(w) = \lambda ||w||_1$.*

**Corollary 3:** *Ridge regression ($l_2$) is equivalent to taking a distortion set $\mathcal{D}(f^{l_2})$ where $f^{l_2}(w) = \lambda ||w||_2$.*

**Corollary 4:** *Elastic net regression [27] ($l_2 + l_1$) is equivalent to taking a distortion set $\mathcal{D}(f^e)$ where $f^e(w) = \lambda_1 ||w||_1 + \lambda_2 ||w||_2^2$, with $\lambda_1 > 0, \lambda_2 > 0$.*

**Corollary 5:** *Group lasso [25] (grouped $l_1$ of $l_2$) is equivalent to taking a distortion set $\mathcal{D}(f^{gl_1})$ where $f^{gl_1}(w) = \sum_{j=1}^m d_j ||w_j||_2$, $d_j$ is the weight for $j$th group and $m$ is the number of group.*

**Corollary 6:** *Local coordinate coding [24] is equivalent to taking a distortion set $\mathcal{D}(f^{lcc})$ where $f^{lcc}(w) = \sum_{i=1}^n |w_i| ||x_i - y||_2^2$, $x_i$ is $i$th basis, $n$ is the number of basis, $y$ is the test example.*

Similar results can be derived for multiple kernel learning [3, 2], overlapped group lasso [16], etc.

### 3.4 Generalization to Other Loss Functions

From the proof of Theorem 1, we can see the Euclidean norm used in Theorem 1 can be generalized to other loss functions too. We only require the loss function is a proper norm in a normed vector space. Thus, we have the following Theorem for a general form of Equation (1).

**Theorem 2.** *Given the relationship function $I_w(\mathcal{X}) = y - Aw$ and $\mathcal{J} \in \mathcal{R}^+$ in a normed vector space, if the loss functional $\mathcal{L}$ is a norm, then Equation (1) is equivalent to the following minimax estimation with a distortion set $\mathcal{D}(\mathcal{J})$:*

$$\min_{w \in \mathcal{R}^p} \max_{\Delta A \in \mathcal{D}(\mathcal{J})} \mathcal{L}(y - (A + \Delta A)w). \tag{7}$$

# 4 Sparse Grouping Representation

## 4.1 Definition of SGR

We consider a classification application where class noise is present. The class noise can be viewed as inter-example distortions. The following novel representation is proposed to deal with such distortions.

**Definition 3.** *Assume all examples are standardized with zero mean and unit variance. Let $\rho_{ij} = \mathbf{x}_i^T \mathbf{x}_j$ be the correlation for any two examples $\mathbf{x}_i, \mathbf{x}_j \in \mathbf{T}$. Given a test example $y$, $w \in \mathcal{R}^n$ is defined as a sparse grouping representation for $y$, if both of the following two conditions are satisfied,*
*(a) If $w_i \geq \epsilon$ and $\rho_{ij} > \delta$, then $|w_i - w_j| \to 0$ (when $\delta \to 1$) for all $i$ and $j$.*
*(b) If $w_i < \epsilon$ and $\rho_{ij} > \delta$, then $w_j \to 0$ (when $\delta \to 1$) for all $i$ and $j$.*
*Especially, $\epsilon$ is the sparsity threshold, and $\delta$ is the grouping threshold.*

This definition requires that if two examples are highly correlated, then the resulted coefficients tend to be identical. Condition (b) produces sparsity by requiring that these small coefficients will be automatically thresholded to zero. Condition (a) preserves grouping effects [27] by selecting all these coefficients which are larger than a certain threshold. In the following we will provide sufficient conditions for the distortion set $\mathcal{D}(\mathcal{J})$ to produce this group level sparsity.

## 4.2 Group Level Sparsity

As known, $\mathcal{D}(l_1)$ or lasso can only select arbitrarily one example from many identical candidates [27]. This leads to the sensitivity to the class noise as the example lasso chooses may be mislabeled. As a consequence, the sparse representation classification (SRC), a lasso based classification schema [21], is not suitable for applications in the presence of class noise. The group lasso can produce group level sparsity, but it uses group label information to restrict the distribution of the coefficients. When there exists group label noise or class noise, group lasso will fail because it cannot correctly determine the group. Definition 3 says that the SGR is defined by example correlations and thus it will not be affected by class noise.

In the general situation where the examples are not identical but have high within-class correlations, we give the following theorem to show that the grouping is robust in terms of data correlation. From now on, for distortion set $\mathcal{D}(f(w))$, we require that $f(w) = 0$ for $w = 0$ and we use a special form of $f(w)$, which is a sum of components $f_j(w)$,

$$f(w) = \mu \sum_{j=1}^{n} f_j(w_j).$$

**Theorem 3.** *Assume all examples are standardized. Let $\rho_{ij} = \mathbf{x}_i^T \mathbf{x}_j$ be the correlation for any two examples. For a given test example $y$, if both $f_i \neq 0$ and $f_j \neq 0$ have first order derivatives, we have*

$$|f_i^{'} - f_j^{'}| \leq \frac{2||y||_2}{\mu} \sqrt{2(1 - \rho_{ij})}. \tag{8}$$

Sketch of the proof: By differentiating $||y - Aw||_2^2 + \sum f_j$ with respect to $w_i$ and $w_j$ respectively, we have $-2\mathbf{x}_i^T \{y - Aw\} + \mu f_i^{'} = 0$ and $-2\mathbf{x}_j^T \{y - Aw\} + \mu f_j^{'} = 0$. The difference of these two equations is $f_i^{'} - f_j^{'} = \frac{2(\mathbf{x}_i^T - \mathbf{x}_j^T)r}{\mu}$ where $r = y - Aw$ is the residual vector. Since all examples are standardized, we have $||\mathbf{x}_i^T - \mathbf{x}_j^T||_2^2 = 2(1 - \rho_{ij})$ where $\rho = \mathbf{x}_i^T \mathbf{x}_j$. For a particular value $w = 0$, we have $||r||_2 = ||y||_2$, and thus we can get $||r||_2 \leq ||y||_2$ for the optimal value of $w$. Combining $r$ and $||\mathbf{x}_i^T - \mathbf{x}_j^T||_2$, we proved the Theorem 3.

This theorem is different from the Theorem 1 in [27] in the following aspects: a) we have no restrictions on the sign of the $w_i$ or $w_j$; b) we use a family of functions which give us more choices to bound the coefficients. As aforementioned, it is not necessary for $f_i$ to be the same with $f_j$ and we even can use different growth rates for different components; and c) $f_i^{'}(w_i)$ does not have to be $w_i$ and a monotonous function with very small growth rate would be enough.

As an illustrative example, we can choose $f_i(w_i)$ or $f_j(w_j)$ to be a second order function with respect to $w_i$ or $w_j$. Then the resulted $|f_i' - f_j'|$ will be the difference of the coefficients $\lambda|w_i - w_j|$ with a constant $\lambda$. If the two examples are highly correlated and $\mu$ is sufficiently large, then we can conclude that the difference of the coefficients will be close to zero.

The sparsity implies an automatic thresholding ability with which all small estimated coefficients will be shrunk to zero, that is, $f(w)$ has to be singular at the point $w = 0$ [9]. Incorporating this requirement with Theorem 3, we can achieve group level sparsity: if some of the group coefficients are small and automatically thresholded to zero, all other coefficients within this group will be reset to zero too. This correlation based group level sparsity does not require any prior information on the distribution of group labels.

To make a good estimator, there are still two properties we have to consider: continuity and un-biasedness [9]. In short, to avoid instability, we always require the resulted estimator for $w$ be a continuous function; and a sufficient condition for unbiasedness is that $f'(|w|) = 0$ when $|w|$ is large. Generally, the requirement of stability is not consistent with that of sparsity. Smoothness determines the stability and singularity at zero measures the degree of sparsity. As an extreme example, $l_1$ can produce sparsity while $l_2$ does not because $l_1$ is singular while $l_2$ is smooth at zero; at the same time, $l_2$ is more stable than $l_1$. More details regarding these conditions can be found in [1, 9].

## 4.3 Sufficient Condition for SGR

Based on the above discussion, we can readily construct a sparse grouping representation based on Equation (5) where we only need to specify a distortion set $\mathcal{D}(f^*(w))$ satisfying the following sufficient conditions:

**Lemma 1: Sufficient condition for SGR.**
(a). $f_j^{*''} \in \mathcal{R}^+$ for all $f_j' \neq 0$.
(b). $f_j^*$ is continuous and singular at zero with respect to $w_j$ for all $j$.
(c). $f_j^{*'}(|w_j|) = 0$ for large $|w_j|$ for all $j$.
Proof: Together with Theorem 3, it is easy to be verified.

As we can see, the regularization term $\lambda l_1 + (1 - \lambda)l_2^2$ proposed by [27] satisfies the above condition (a) and (b), but it fails to comply with (c). So, it may become biased for large $|w|$. Based on these conditions, we can easily construct regularization terms $f^*$ to generate the sparse grouping representation. We will call these $f^*$ as core functions for producing the SGR. As some concrete examples, we can construct a large family of clipped $\mu_1 L_q + \mu_2 l_2^2$ where $0 < q \leq 1$ by restricting $f_i' = w_i I(|w_i| < \epsilon) + c$ for some constant $\epsilon$ and $c$. Also, SCAD [9] satisfies all three conditions so it belongs to $f^*$. This gives more theoretic justifications for previous empirical success of using SCAD.

## 4.4 Generalization Bounds for Presence of Class Noise

We will follow the algorithm given in [21] and merely replace the lasso with the SGR or group lasso. After estimating the (minimax) optimal combining coefficient vector $w^*$ by the SGR or group lasso, we may calculate the distance from the new test data $y$ to the projected point in the subspace spanned by class $C_i$:

$$d_i(A, w^*|_{C_i}) = d_i(A|_{C_i}, w^*) = ||y - Aw^*|_{C_i}||_2 \tag{9}$$

where $w^*|_{C_i}$ represents restricting $w^*$ to the $i$th class $C_i$; that is, $(w^*|_{C_i})_j = w_j^* 1(x_j \in C_i)$, where $1(\cdot)$ is an indicator function; and similarly $A|_{C_i}$ represents restricting $A$ to the $i$th class $C_i$.

A decision rule may be obtained by choosing the class with the minimum distance:

$$\hat{i} = \text{argmin}_{i \in \{1, \cdots, M\}} \{d_i\}. \tag{10}$$

Based on these notations, we now have the following generalization bounds for the SGR in the presence of class noise in the training data.

**Theorem 4.** *All examples are standardized to be zero mean and unit variance. For an arbitrary class $C_i$ of $N$ examples, we have $p$ ($p < 0.5$) percent (fault level) of labels mis-classified into class*

$C_k \neq C_i$. *We assume $w$ is a sparse grouping representation for any test example $y$ and $\rho_{ij} > \delta$ ($\delta$ is in Definition 3) for any two examples. Under the distance function $d(A|_{C_i}, w) = d(A, w|_{C_i}) = ||y - Aw|_{C_i}||_2$ and $f'_j = w$ for all $j$, we have confidence threshold $\tau$ to give correct estimation $\hat{i}$ for $y$, where*

$$\tau \leq \frac{(1 - p) \times N \times (w^0)^2}{d},$$

*where $w^0$ is a constant and the confidence threshold is defined as $\tau = d_i(A|_{C_i}) - d_i(A|_{C_k})$.*

Sketch of the proof: Assume $y$ is in class $C_i$. The correctly labeled (mislabeled, respectively) subset for $C_i$ is $C_i^1$ ($C_i^2$, respectively) and the size of set $C_i^1$ is larger than that of $C_i^2$. We use $A^1 w$ to denote $Aw|_{C_i^1}$ and $A^2 w$ to denote $Aw|_{C_i^2}$. By triangular inequality, we have

$$\tau = ||y - Aw|_{C_i^1}||_2 - ||y - Aw|_{C_i^2}||_2$$
$$\leq ||A^1 w - A^2 w||_2.$$

For each $k \in C_i^1$, we differentiate with respect to $w_k$ and do the same procedure as in proof of Theorem 3. Then summarizing all equalities for $C_i^1$ and repeating the same procedure for each $i \in C_i^2$. Finally we subtract the summation of $C_i^2$ from the summation of $C_i^1$. Use the conditions that $w$ is a sparse grouping representation and $\rho_{ij} > \delta$, combing Definition 3, so all $w_k$ in class $C_i$ should be the same as a constant $w^0$ while others $\rightarrow 0$. By taking the $l_2$-norm for both sides, we have $||A^1 w - A^2 w||_2 \leq \frac{(1-p)N(w^0)^2}{d}$.

This theorem gives an upper bound for the fault-tolerance against class noise. By this theorem, we can see that the class noise must be smaller than a certain value to guarantee a given fault correction confidence level $\tau$.

## 5  Experimental Verification

In this section, we compare several methods on a challenging low-resolution face recognition task (multi-class classification) in the presence of class noise. We use the Yale database [4] which consists of 165 gray scale images of 15 individuals (each person is a class). There are 11 images per subject, one per different facial expression or configuration: center-light, w/glasses, happy, left-light, w/no glasses, normal, right-light, sad, sleepy, surprised, and wink. Starting from the orignal $64 \times 64$ images, all images are down-sampled to have a dimension of $49$. A training/test data set is generated by uniformly selecting 8 images per individual to form the training set, and the rest of the database is used as the test set; repeating this procedure to generate five random split copies of training/test data sets. Five class noise levels are tested. Class noise level=$p$ means there are $p$ percent of labels (uniformly drawn from all labels of each class) mislabeled for each class.

For SVM, we use the standard implementation of multiple-class (one-vs-all) LibSVM in MatlabArsenal[1]. For lasso based SRC, we use the CVX software [13, 14] to solve the corresponding convex optimization problems. The group lasso based classifier is implemented in the same way as the SRC. We use a clipped $\lambda l_1 + (1 - \lambda) l_2$ as an illustrative example of the SGR, and the corresponding classifier is denoted as SGRC. For lasso, group Lasso and the SGR based classifier, we run through $\lambda \in \{0.001, 0.005, 0.01, 0.05, 0.1, 0.2\}$ and report the best results for each classifier. Figure 1 (b) shows the parameter range of $\lambda$ that is appropriate for lasso, group lasso and the SGR based classifier. Figure 1 (a) shows that the SGR based classifier is more robust than lasso or group lasso based classifier in terms of class noise. These results verify that in a novel application when there exists class noise in the training data, the SGR is more suitable than group lasso for generating group level sparsity.

## 6  Conclusion

Towards a better understanding of various regularized procedures in robust linear regression, we introduce a robust minimax framework which considers both additive and multiplicative noise or distortions. Within this unified framework, various regularization terms correspond to different

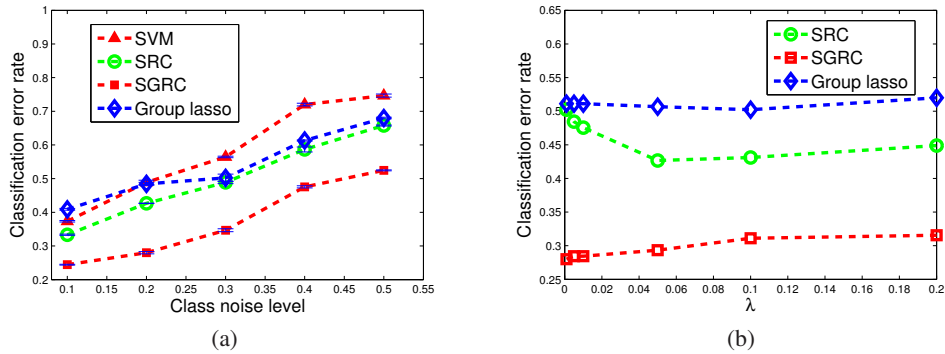

Figure 1: (a) Comparison of SVM, SRC (lasso), SGRC and Group lasso based classifiers on the low resolution Yale face database. At each level of class noise, the error rate is averaged over five copies of training/test datasets for each classifier. For each classifier, the variance bars for each class noise level are plotted. (b) Illustration of the paths for SRC (lasso), SGRC and group lasso. $\lambda$ is the weight for regularization term. All data points are averaged over five copies with the same class noise level of 0.2.

distortions to the original data matrix. We further investigate a novel sparse grouping representation (SGR) and prove sufficient conditions for generating such group level sparsity. We also provide a generalization bound for the SGR. In a novel classification application when there exists class noise in the training example, we show that the SGR is more robust than group lasso. The SCAD and clipped elastic net are special instances of the SGR.

## Footnotes

[1]A matlab package for classification algorithms which can be downloaded from http://www.informedia.cs.cmu.edu/yanrong/MATLABArsenal/MATLABArsenal.htm.

# References

[1] A. Antoniadis and J. Fan. Regularitation of wavelets approximations. *J. the American Statistical Association*, 96:939–967, 2001.

[2] F. Bach. Consistency of the group lasso and multiple kernel learning. *Journal of Machine Learning Research*, 9:1179–1225, 2008.

[3] F. Bach, G. R. G. Lanckriet, and M. I. Jordan. Multiple kernel learning, conic duality, and the smo algorithm. In *Proceedings of the Twenty-first International Conference on Machine Learning*, 2004.

[4] P. N. Bellhumer, J. Hespanha, and D. Kriegman. Eigenfaces vs. fisherfaces: Recognition using class specific linear projection. *IEEE Trans. Pattern Anal. Mach. Intelligence*, 17(7):711–720, 1997.

[5] L. Breiman. Heuristics of instability and stabilization in model selection. *Ann. Statist.*, 24:2350–2383, 1996.

[6] E. Candés, J. Romberg, and T. Tao. Stable signal recovery from incomplete and inaccurate measurements. *Comm. on Pure and Applied Math*, 59(8):1207–1233, 2006.

[7] E. Candés and T. Tao. Near-optimal signal recovery from random projections: Universal encoding strategies? *IEEE Trans. Information Theory*, 52(12):5406–5425, 2006.

[8] D. Donoho. For most large underdetermined systems of linear equations the minimum l1 nom solution is also the sparsest solution. *Comm. on Pure and Applied Math*, 59(6):797–829, 2006.

[9] J. Fan and R. Li. Variable selection via nonconcave penalized likelihood and its oracle properties. *J. Am. Statist. Ass.*, 96:1348–1360, 2001.

[10] I. Frank and J. Friedman. A statistical view of some chemometrics regression tools. *Technometrics*, 35:109–148, 1993.

[11] L. El Ghaoui and H. Lebret. Robust solutions to least-squares problems with uncertain data. *SIAM Journal Matrix Analysis and Applications*, 18:1035–1064, 1997.

[12] G.H. Golub and C.F. Van Loan. *Matrix computations*. Johns Hopkins Univ Pr, 1996.

[13] M. Grant and S. Boyd. Graph implementations for nonsmooth convex programs, recent advances in learning and control. *Lecture Notes in Control and Information Sciences*, pages 95–110, 2008.

[14] M. Grant and S. Boyd. UCI machine learning repositorycvx: Matlab software for disciplined convex programming, 2009.

[15] A. Hoerl and R. Kennard. Ridge regression. *Encyclpedia of Statistical Science*, 8:129–136, 1988.

[16] L. Jacob, G. Obozinski, and J.-P. Vert. Group lasso with overlap and graph lasso. In *Proceedings of the Twenty-six International Conference on Machine Learning*, pages 433–440, 2009.

[17] J. Maletic and A. Marcus. Data cleansing: Beyond integrity analysis. In *Proceedings of the Conference on Information Quality*, 2000.

[18] K. Orr. Data quality and systems theory. *Communications of the ACM*, 41(2):66–71, 1998.

[19] R. Tibshirani. Regression shrinkage and selection via the lasso. *J. R. Statist. Soc. B*, 58:267–288, 1996.

[20] R. Tibshirani, M. Saunders, S. Rosset, J. Zhu, and K. Knight. Sparsity and smoothness via the fused lasso. *J.R.Statist.Soc.B*, 67:91–108, 2005.

[21] J. Wright, A.Y. Yang, A. Ganesh, S.S. Sastry, and Y. Ma. Robust face recognition via sparse representation. *IEEE Transactions on Pattern Analysis and Machine Intelligence*, pages 210–227, 2009.

[22] X. Wu. *Knowledge Acquisition from Databases*. Ablex Pulishing Corp, Greenwich, CT, USA, 1995.

[23] H. Xu, C. Caramanis, and S. Mannor. Robust regression and lasso. In *NIPS*, 2008.

[24] K. Yu, T. Zhang, and Y. Gong. Nonlinear learning using local coordinate coding. In *Advances in Neural Information Processing Systems*, volume 22, 2009.

[25] M. Yuan and Y. Lin. Model selection and estimation in regression with grouped variables. *Journal of The Royal Statistical Society Series B*, 68(1):49–67, 2006.

[26] X. Zhu, X. Wu, and S. Chen. Eliminating class noise in large datasets. In *Proceedings of the 20th ICML International Conference on Machine Learning*, Washington D.C., USA, March 2003.

[27] H. Zou and T. Hastie. Regularization and variable selection via the elastic net. *J. R. Statist. Soc. B*, 67(2):301–320, 2005.

